# On the asymptotic equivalence between differential Hebbian and temporal difference learning using a local third factor

**Christoph Kolodziejski**[1,2], **Bernd Porr**[3], **Minija Tamosiunaite**[1,2,4], **Florentin Wörgötter**[1,2]

[1] Bernstein Center for Computational Neuroscience Göttingen
[2] Georg-August University Göttingen, Department of Nonlinear Dynamics
Bunsenstr. 10, 37073 Göttingen, Germany
[3] University of Glasgow, Department of Electronics & Electrical Engineering
Glasgow, GT12 8LT, Scotland
[4] Vytautas Magnus University, Department of Informatics
Vileikos 8, 44404, Kaunas, Lithuania
`kolo|minija|worgott@bccn-goettingen.de, b.porr@elec.gla.ac.uk`

## Abstract

In this theoretical contribution we provide mathematical proof that two of the most important classes of network learning - correlation-based differential Hebbian learning and reward-based temporal difference learning - are asymptotically equivalent when timing the learning with a local modulatory signal. This opens the opportunity to consistently reformulate most of the abstract reinforcement learning framework from a correlation based perspective that is more closely related to the biophysics of neurons.

## 1  Introduction

The goal of this study is to prove that the most influential form of reinforcement learning (RL) [1], which relies on the temporal difference (TD) learning rule [2], is equivalent to correlation based learning (Hebb, CL) which is convergent over wide parameter ranges when using a local third factor, as a gating signal, together with a differential Hebbian emulation of CL.

Recently there have been several contributions towards solving the question of equivalence of different rules [3, 4, 5, 6], which presented specific solutions to be discussed later (see section 4). Thus, there is more and more evidence emerging that Hebbian learning and reinforcement learning can be brought together under a more unifying framework. Such an equivalence would have substantial influence on our understanding of network learning as these two types of learning could be interchanged under these conditions.

The idea of differential Hebbian learning was first used by Klopf [7] to describe classical conditioning relating to the stimulus substitution model of Sutton [8]. One of its most important features is the implicit introduction of negative weight changes (LTD), which leads to intrinsic stabilization properties in networks. Earlier approaches had to explicitly introduce negative weight changes into the learning rule, e.g. by ways of a threshold [9].

One drawback of reinforcement learning algorithms, like temporal difference learning, is their use of discrete time and discrete non-overlapping states. In real neural systems, time is continuous and the state space can only be represented by the activity of neurons, many of which will be active at the same time and for the same "space". This creates a rather continuous state space representation in real systems. In order to allow for overlapping states or for generalizing over a wider range of input regions, RL algorihtms are usually extended by value function approximation methods [1].

However, while biologically more realistic [10], this makes initially elegant RL algorithms often quite opaque and convergence can many times not be guaranteed anymore [11]. Here we are not concerned with function approximation, but instead address the question of how to transform an RL algorithm (TD-learning) to continuous time using differential Hebbian learning with a local third factor and remaining fully compatible with neuronally plausible operations.

Biophysical considerations about how such a third factor might be implemented in real neural tissue are of secondary importance for this study. At this stage we are concerned with a formal proof only.

## 1.1 Emulating RL by Temporal Difference Learning

Reinforcement learning maximizes the rewards $r(s)$ an agent will receive in the future when following a policy $\pi$ traveling along states $s$. The return $R$ is defined as the sum of the future rewards: $R(s_i) = \sum_k \gamma^k r(s_{i+k+1})$, where future rewards are discounted by a factor $0 < \gamma \leq 1$. One central goal of RL is to determine the values $V(s)$ for each state given by the average expected return $E^\pi\{R\}$, that can be obtained when following policy $\pi$. Many algorithms exist to determine the values, almost all of which rely on the temporal difference (TD) learning rule (Eq. 1) [2].

Every time the agent encounters a state $s_i$, it updates the value $V(s_i)$ with the discounted value $V(s_{i+1})$ and the reward $r(s_{i+1})$ of the next state that is associated with the consecutive state $s_{i+1}$:

$$V(s_i) \rightarrow (1 - \alpha)V(s_i) + \alpha(r(s_{i+1}) + \gamma V(s_{i+1})) \tag{1}$$

where $\alpha$ is the learning rate. This rule is called TD($\lambda = 0$), short TD(0), as it only evaluates adjacent states. For values of $\lambda \neq 0$ more of the recently visited states are used for value-function update. TD(0) is by far the most influential RL learning rule as it is the simplest way to assure optimality of learning [12, 1].

## 1.2 Differential Hebbian learning with a local third factor

In traditional Hebbian learning, the change of a weight $\rho$ relies on the correlation between input $u(t)$ and output $v(t)$ of a neuron: $\rho'(t) = \tilde{\alpha} \cdot u(t) \cdot v(t)$, where $\tilde{\alpha}$ is the learning rate and prime denotes the temporal derivative. If we consider the *change* of the post-synaptic signal and, therefore, replace $v(t)$ with $v'(t)$, we will arrive at differential Hebbian learning. Then, also negative weight changes are possible and this yields properties similar to experimental neurophysiological observations (spike-timing dependent plasticity, [13]).

In order to achieve the equivalence (see section 4 for a discussion) we additionally introduce a local third modulatory factor $M_k(t)$ responsible for controlling the learning [14]. Here local means that each input $u_k(t)$ controls a separate third factor $M_k(t)$ which in turn modulates only the weight change of the corresponding weight $\rho_k(t)$. The local three-factor differential-Hebbian learning rule is then:

$$\rho_k'(t) = \tilde{\alpha} \cdot u_k(t) \cdot v'(t) \cdot M_k(t) \tag{2}$$

where $u_k(t)$ is the considered pre-synaptic signal and

$$v(t) = \sum_n \rho_n(t)u_n(t) \tag{3}$$

the post-synaptic activity of a model neuron with weights $\rho_n(t)$. We will assume in the following that our modulatory signal $M_k(t)$ is either 1 or 0, thus represented by a step function.

## 2 Analytical derivation

We are going to analyze the weight change of weight $\rho_i(t)$ when considering two consecutive signals $u_i(t)$ and $u_{i+1}(t)$ with the index $i$ representing a temporal (and not e.g. a spatial) ordering. The local third factor $M_i(t)$ opens a time window for its corresponding weight $\rho_i(t)$ in which changes can occur. Although this time window could be located anywhere depending on the input $u_i(t)$ it should be placed at the end of the state $s_i(t)$ as it makes only sense if states correlate with their successor.

The relation between state $s(t)$ and input $u(t)$ is determined by a convolution: $u(t) = \int_0^\infty s(z)h(t - z)dz$ with filter function $h(t)$ which are identical for all states. As we are using only states that are

either on or off during a visiting duration $S$, the input functions $u(t)$ do not differ between states. Therefore we will use $u_i(t)$ (with index $i$) having a particular state in mind and $u(t)$ (without index $i$) when pointing to functional development.

Furthermore we define the time period between the end of a state $s_i(t)$ and the beginning of the next state $s_{i+1}(t)$ as $T$ ($T < 0$ in case of overlapping states). Concerning the modulatory third factor $M_i(t)$ we define its length as $L$, and the time period between beginning of $M_i(t)$ and the end of the corresponding state $s_i(t)$ as $O$. These four parameters ($L$, $O$, $T$, and $S$) are constant over states and are displayed in detail in fig. 1 B.

## 2.1 Analysis of the differential equation

For the following analysis we need to substitute Eq. 3 in Eq. 2 and solve this differential equation which consists of a homogeneous and an inhomogeneous part:

$$\rho_i'(t) = \tilde{\alpha} \cdot M_i(t) \cdot u_i(t)[u_i(t) \cdot \rho_i(t)]' + \tilde{\alpha} \cdot M_i(t) \cdot u_i(t)[\sum_{j \neq i} u_j(t) \cdot \rho_j(t)]' \tag{4}$$

where the modulator $M_i(t)$ is defining the integration boundaries. The first summand leads us to the homogeneous solution which we will define as auto-correlation $\rho^{ac}(t)$. The second summand(s) on the other hand will lead to the inhomogeneous solution and this we will define as cross-correlation $\rho^{cc}(t)$. Together we have $\rho(t) = \rho^{ac}(t) + \rho^{cc}(t)$.

In general the overall change of the weight $\rho_i(t)$ after integrating over the visiting duration of $s_i(t)$ and $s_{i+1}(t)$ and using the modulatory signal $M_i(t)$ is: $\Delta\rho_i =: \Delta_i = \Delta_i^{ac} + \Delta_i^{cc}$

Without restrictions, we can now limit further analysis of Eq. 4, in particular of the cross-correlation term, to the case of $j = i+1$ as the modulatory factor only effects the weight of the following state.

Since weight changes are in general slow, we can assume a quasi-static process ($\frac{\rho_i'}{\rho_i} \ll \frac{u_i'}{u_i}$, $\alpha \to 0$). As a consequence, the derivatives of $\rho$ on the right hand side of Eq. 4 can be neglected.

The solution of the auto-correlation $\rho_i^{ac}(t)$ is then in general:

$$\rho_i^{ac}(t) = \rho_i^{ac}(t_0)e^{\tilde{\alpha} \cdot M_i(t) \cdot \frac{1}{2}[u_i^2(t) - u_i^2(t_0)]} \tag{5}$$

and the overall weight change with the third factor being present between $t = O + S$ and $t = O + S + L$ (fig. 1 B) is therefore:

$$\Delta_i^{ac} = \rho_i(e^{\tilde{\alpha}\frac{1}{2}[u_i^2(O+S+L) - u_i^2(O+S)]} - 1) \tag{6}$$

Using again the argument of a quasi-static process ($\tilde{\alpha} \to 0$), we can expand the exponential function to the first order:

$$\Delta_i^{ac}: \quad = \quad -\tilde{\alpha}\rho_i\frac{1}{2}[u_i^2(O+S) - u_i^2(O+S+L) + o(\tilde{\alpha})] \tag{7}$$

$$= \quad -\tilde{\alpha}\rho_i\kappa \tag{8}$$

where we have defined $\kappa$ in the following way:

$$\kappa(L, O, S) \quad = \quad \frac{1}{2}[u^2(O+S) - u^2(O+S+L) + o(\tilde{\alpha})] \tag{9}$$

which is independent of $i$ since we assume all state signals as identical.

Next we investigate the cross-correlation $\rho^{cc}(t)$ again under the assumption of a quasi-static process. This leads us to:

$$\rho_i^{cc}(t) = \rho_i^{cc}(t_0) + \tilde{\alpha}\rho_{i+1}\int_0^t M_i(z) \cdot u_i(z)u_{i+1}'(z)dz \tag{10}$$

which yields assuming a time shift between signals $u_i$ and $u_{i+1}$ of $S+T$, i.e. $u_i(t-S-T) = u_{i+1}(t)$ an overall weight change of

$$\Delta_i^{cc} = \tilde{\alpha}\rho_{i+1}\int_{O+S}^{O+S+L} u_i(z)u_i'(z-S-T)dz := \tilde{\alpha}\rho_{i+1}\tau \tag{11}$$

whereas the third factor was being present between $t = O + S$ and $t = O + S + L$ (fig. 1 B). Additionally we defined $\tau$ as follows:

$$\tau(L, O, T, S) = \int_{O-T}^{O+L-T} u(z + S + T)u'(z)dz \tag{12}$$

which, too, is independent of $i$.

Both $\tau$ and $\kappa$ depend on the actually used signal shape $u(t)$ and the values for the parameters $L$, $O$, $T$ and $S$.

## 2.2 Analysis of the network

After the analysis of the auto- and cross-correlation of Eq. 4 we are going to discuss the weight changes in a network context with a reward only at the terminal state (non-terminal reward states will be discussed in section 4). Without restrictions, we can limit this discussion to the situation in Fig. 1 A where we have one intermediate state transition (from $s_i$ to $s_{i+1}$) and a final one (from $s_{i+1}$ to $s_R$) which yields a reward. The weight associated with the reward state $s_R$ is set to a constant value unequal to zero.

Therefore three-factor differential Hebbian will influence two synaptic connections $\rho_i$ and $\rho_{i+1}$ of states $s_i$ and $s_{i+1}$ respectively, which directly project onto neuron $v$.

Fig. 1 B shows a realistic situation of state transitions leaving the old state $s_{i-1}$ and entering the new state $s_i$ and so on. The signals as such could be considered as membrane voltages or firing rates of neurons.

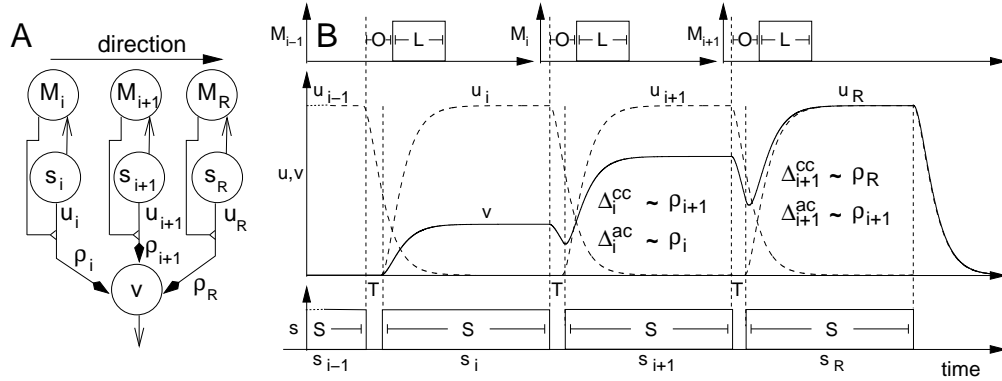

Figure 1: The setup is shown in panel A and the signal structure in panel B. (A) Three states, including the rewarded state, converge on the neuron which learns according to Eq. 2. Each state $s_i$ controls the occurrence of the modulatory factor $M_i$ which in turn will influence learning at synapse $\rho_i$. The states $s$ will be active according to the direction arrow. (B) The lower part shows the states $s_i$ which have a duration of length $S$. We assume that the duration for the transition between two states is $T$. In the middle the output $v$ and the signals $u$ are depicted. Here $u$ is given by $u(t) = \int_0^S (e^{-a(t-z)} - e^{-b(t-z)}) dz$. The third factor $M_i$ is released for the duration $L$ after a time delay of $O$ and is shown in the upper part. For each state the weight change separated into auto-correlation $\Delta^{ac}$ and cross-correlation $\Delta^{cc}$ and their dependence on the weights according to Eq. 7 and 11 are indicated.

We will start our considerations with the weight change of $\rho_i$ which is only influenced by the visiting state $s_i$ itself and by the transition between $s_i$ and $s_{i+1}$. The weight change $\Delta_i^{ac}$ caused by the auto-correlation ($s_i$ with itself) is governed by the weight $\rho_i$ of state $s_i$ (see Eq. 8) and is negative as the signal $u_i$ at the the end of the state decays ($\kappa$ is positive, though, because we factorized a minus sign from Eq. 6 to Eq 7). The cross-correlation ($\Delta_i^{cc}$), however, is proportional to the weight $\rho_{i+1}$ of the following state $s_{i+1}$ (see Eq. 11) and is positive because the positive derivative of the next state signal $u_{i+1}$ correlates with the signal $u_i$ of state $s_i$. According to these considerations the contributions for the $\Delta_{i+1}$-values can be discussed in an identical way for the following sequence $(s_{i+1}, s_R)$.

In general the weight after a single trial is the sum of the old weight $\rho_i$ with the two $\Delta$-values:

$$\rho_i \rightarrow \rho_i + \Delta_i^{ac} + \Delta_i^{cc} \tag{13}$$

Using Eq. 8 and Eq. 11 we can reformulate Eq. 13 into

$$\rho_i \rightarrow \rho_i - \tilde{\alpha} \cdot \kappa \cdot \rho_i + \tilde{\alpha} \cdot \tau \cdot \rho_{i+1} \tag{14}$$

Substituting $\alpha = \tilde{\alpha} \cdot \kappa$ and $\gamma = \tau/\kappa$ we get

$$\rho_i \rightarrow (1 - \alpha) \cdot \rho_i + \alpha \cdot \gamma \cdot \rho_{i+1} \tag{15}$$

At this point we can make the transition from weights $\rho_i$ (differential Hebbian learning) to states $V(s_i)$ (temporal difference learning). Additionally we note that sequences only terminate at $i + 1$, thus this index will capture the reward state $s_R$ and its value $r(s_{i+1})$, while this is not the case for all other indices (see section 4 for a detail discussion of rewards at non-terminal states). Consequently this gives us an equation almost identical to Eq 1:

$$V(s_i) \rightarrow (1 - \alpha)V(s_i) + \alpha \cdot \gamma[r(s_{i+1}) + V(s_{i+1})] \tag{16}$$

where one small difference arises as in Eq. 16 the reward is scaled by $\gamma$. However, this has no influence as numerical reward values are arbitrary. Thus, if learning follows this third factor differential Hebbian rule, weights will converge to the optimal estimated TD-values. This proves that, under some conditions for $\kappa$ and $\tau$ (see below), TD(0) and the here proposed three factor differential Hebbian learning are indeed asymptotically equivalent.

## 2.3 Analysis of $\kappa$ and $\gamma$

Here we will take a closer look at the signal shape and the parameters ($L$, $O$, $T$ and $S$) which influence the values of $\kappa$ (Eq. 9) and $\tau$ (Eq. 12) and therefore $\gamma = \tau/\kappa$. For guaranteed convergence these values are constraint by two conditions, $\tau \geq 0$ and $\kappa > 0$ (where $\kappa = 0$ is allowed in case of $\tau = 0$), which come from Eq. 14. A non-positive value of $\kappa$ would lead to divergent weights $\rho$ and a negative value of $\tau$ to oscillating weight pairs $(\rho_i, \rho_{i+1})$. However even if fulfilled, these conditions will not always lead to meaningful weight developments. A $\tau$-value of 0 leaves all weights at their initial weight value and discount factors, which are represented by $\gamma$-values exceeding 1, are usually not considered in reinforcement learning [1]. Thus it makes sense to introduce more rigorous conditions and demand that $0 < \gamma \leq 1$ and $\kappa > 0$.

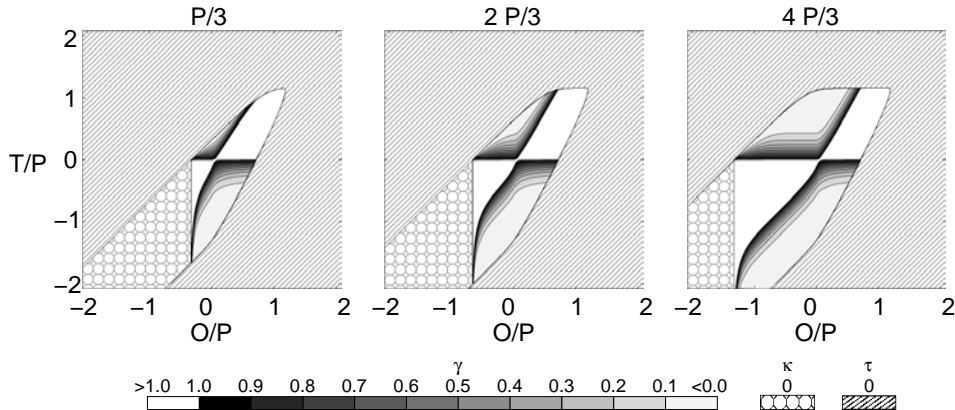

Figure 2: Shown are $\gamma$-values dependent on the ratio $O/P$ and $T/P$ for three different values of $L/P$ (1/3, 2/3, and 4/3). Here $P$ is the length of the rising as well as the falling phase. The shape of the signal $u$ is given by $u(t) = \int_0^S (e^{-a\,(t-z)} - e^{-b\,(t-z)})\,dz$ with parameters $a = 0.006$ and $b = 0.066$. The individual figures are subdivided into a patterned area where the weights will diverge ($\kappa = 0$, see Eq.7), a striped area where no overlap between both signals and the third factor exists and into a white area that consists of $\gamma$-values which, however, are beyond a meaningful range ($\gamma > 1$). The detailed gray shading represent $\gamma$-values ($0 < \gamma \leq 1$) for which convergence is fulfilled.

Furthermore, as these conditions depend on the signal shape, the following theoretical considerations need to be guided by biophysics. Hence, we will discuss neuronally plausible signals that can arise at a synapse. This constrains $u$ to functions that posses only one maximum and divide the signal into a rising and a falling phase.

One quite general possibility for the shape of the signal $u$ is the function used in Fig. 1 for which we investigate the area of convergence. As we have three (we do not have to consider the parameter $S$ if we take this value to be large compared to $|T|$, $L$ or $O$) parameters to be varied, Fig. 2 shows the $\gamma$-value in 3 different panels. In each panel we varied the parameters $O$ and $T$ from minus to plus $2P$ where $P$ is the time the signal $u$ needs to reach the maximum. In each of the panels we plot $\gamma$-values for a particular value of $L$.

Regarding $\kappa$ the condition formed by Eq. 9 for the shape of the signal $u(t)$ is in general already fulfilled by using neuronally plausible signals and the third factor at the end of each state. As the signals start to decay after the end of a state visit, $u(O + S)$ is always larger than $u(O + S + L)$ and therefore $\kappa > 0$. Only if the third factor is shifted (due to the parameter $O$, see fig. 1 B for more details) to regions of the signal $u$ where the decay has not yet started ($O < -L$) or has already ended ($O > P$) the difference of $u(O+S)$ and $u(O+S+L)$ is 0 which leads using Eq. 9 to $\kappa = 0$. This is indicated by the patterned area in fig. 2.

A gray shading displays in detail the $\gamma$-values for which the condition is fulfilled, whereas white represents those areas for which we receive $\gamma > 1$. The striped area indicates parameter configurations for which no overlap between two consecutive signals and the third factor exist ($\tau = 0$).

The different frames show clearly that the area of convergence changes only gradually and the area as such is increasing with increasing duration of the third factor. Altogether it shows that for a general neuronally plausible signal shape $u$ the condition for asymptotic equivalence between temporal difference learning and differential Hebbian learning with a local third factor is fulfilled for a wide parameter range.

## 3 Simulation of a small network

In this section we show that we can reproduce the behavior of TD-learning in a small linear network with two terminal states. This is done with a network of neurons designed according to our algorithm with a local third factor. Obtained weights of the differential Hebbian learning neuron represent the corresponding TD-value (see fig. 3 A). It is known that in a linear TD-learning system with two terminal states (one is rewarded, the other not) and a $\gamma$-value close to 1, values at the end of learning will represent the probability of reaching the reward state starting at the corresponding state (compare [1]). This is shown, including the weight development, in panel (B).

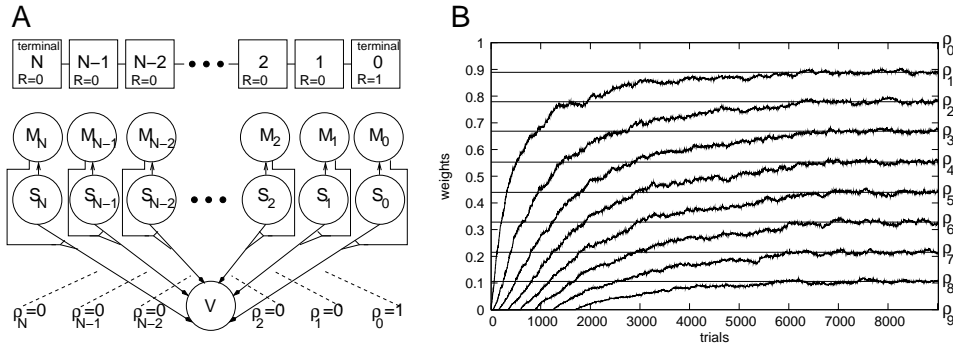

Figure 3: The linear state arrangement and the network architecture is shown in panel A. The corresponding weights after a typical experiment are depicted in panel B. The lines represent the mean of the last 2000 weight-values of each state and are distributed uniformly (compare [1]). The signal shape is given by $u(t) = \int_0^S (e^{-a(t-z)} - e^{-b(t-z)})\, dz$ with parameters $a = 0.006$ and $b = 0.066$. Furthermore is $O = 1/20\,P$, $L = P$, $T = 0$ (which yields $\gamma \simeq 1$), $N = 9$, and $\tilde{\alpha} = 0.01$.

# 4 Discussion

The TD-rule has become the most influential algorithm in reinforcement learning, because of its tremendous simplicity and proven convergence to the optimal value function [1]. It had been successfully transferred to control problems, too, in the form of Q- or SARSA learning [15, 16], which use the same algorithmic structure, while maintaining similar advantageous mathematical properties [15].

In this study we have shown that TD(0)-learning and differential Hebbian learning modulated by a local third factor are equivalent under certain conditions. This proof relies only on commonly applicable, fairly general assumptions, thus rendering a generic result not constraining the design of larger networks. However, in which way the timing of the third factor is implemented in networks will be an important issue when constructing such networks.

Several earlier results have pointed to the possibility of an equivalence between RL and CL. Izhikevich [3] solved the distal reward problem using a spiking neural network, yet with fixed exponential functions [17] to emulate differential Hebbian characteristics. His approach is related to neurophysiologically findings on spike-timing dependent plasticity (STDP, [13]). Each synapse learned the correlation between conditioned stimuli and unconditioned stimuli (e.g. a reward) through STDP and a third signal. Furthermore Roberts [4] showed that that asymmetrical STDP and temporal difference learning are related. In our differential Hebbian learning model, in contrast to the work described above, STDP emerges automatically because of the use of the derivative in the postsynaptic potential (Eq. 2). Rao and Sejnowski [18] showed that using the temporal difference will directly lead to STDP, but they could not provide a rigorous proof for the equivalence. Recently, it has been shown that the online policy-gradient RL-algorithm (OLPOMDP, [19]) can be emulated by spike timing dependent plasticity [5], however, in a complex way using a global reward signal. On the other hand, the observations reported here provide a rather simple, equivalent correlation based implementation of TD and support the importance of three factor learning for providing a link between conventional Hebbian approaches and reinforcement learning.

In most physiological experiments [20, 21, 22] the reward is given at the end of the stimulus sequence. Our assumption that the reward state is a terminating state and is therefore only at the end of the learning sequence conforms, thus, to this paradigm. However, for TD in general we cannot assume that the reward is only provided at the end. Differential Hebbian learning will then lead to a slightly different solution compared to TD-learning. This solution has already been discussed in a another context [23]. Specifically, the difference in our case is the final result for the state-value after convergence for states that provide a reward: We get $V(s) \rightarrow \gamma V(s_{i+1}) + r(s_{i+1}) - r(s_i)$ compared to TD learning: $V(s) \rightarrow \gamma V(s_{i+1}) + r(s_{i+1})$. It would be interesting to assess with physiological and or behavioral experiments, which of the two equations does more closely represent experimental reality.

Our results rely in a fundamental way on the third factor $M_i$, and the analysis performed in this study indicates that the third factor is necessary for the emulation of TD-learning by a differential Hebb rule. To explain the reason for this requires a closer look at the temporal difference learning rule. We find that the TD-rule requires a leakage term $-\alpha \cdot V(s)$. If this term does not exist, values would diverge. It has been shown [24] that in differential Hebbian learning without a third factor, however, the auto-correlation part, which is the source of the leakage needed, (see Eq. 13 and Eq. 7) is non existing. This shows that just through a well-timed third factor the ratio between cross-correlation and auto-correlation term is correctly adjusted. This ratio is at the end responsible for the $\gamma$-value we will get using differential Hebbian learning to emulate TD-learning.

# References

[1] R.S. Sutton and A.G. Barto. *Reinforcement Learning: An Introduction*. MIT Press, Cambridge, MA, 1998.

[2] R. S. Sutton. Learning to predict by the method of temporal differences. *Mach. Learn.*, 3:9–44, 1988.

[3] E. Izhikevich. Solving the distal reward problem through linkage of stdp and dopamine signaling. *Cereb. Cortex.*, 17:2443–2452, 2007.

[4] PD. Roberts, RA. Santiago, and G. Lafferriere. An implementation of reinforcement learning based on spike-timing dependent plasticity. *Biol. Cybern.*, in press.

[5] R. V. Florian. Reinforcement learning through modulation of spike-timing-dependent synaptic plasticity. *Neural Comput.*, 19:1468–1502, 2007.

[6] W. Potjans, A. Morrison, and M. Diesmann. A spiking neural network model of an actor-critic learning agent. *Neural Comput.*, 21:301–339, 2009.

[7] A. H. Klopf. A neuronal model of classical conditioning. *Psychobiol.*, 16(2):85–123, 1988.

[8] R. Sutton and A. Barto. Towards a modern theory of adaptive networks: Expectation and prediction. *Psychol. Review*, 88:135–170, 1981.

[9] E. Oja. A simplified neuron model as a principal component analyzer. *J. Math. Biol.*, 15(3):267–273, 1982.

[10] M. Tamosiunaite, J. Ainge, T. Kulvicius, B. Porr, P. Dudchenko, and F. Wörgötter. Path-finding in real and simulated rats: On the usefulness of forgetting and frustration for navigation learning. *J. Comp. Neurosci.*, 25(3):562–582, 2008.

[11] M. Wiering. Convergence and divergence in standard averaging reinforcement learning. In J Boulicaut, F Esposito, F Giannotti, and D Pedreschi, editors, *Proceedings of the 15th European Conference on Machine learning ECML'04*, pages 477–488, 2004.

[12] P. Dayan and T. Sejnowski. Td($\lambda$) converges with probability 1. *Mach. Learn.*, 14(3):295–301, 1994.

[13] H. Markram, J. Lübke, M. Frotscher, and B. Sakmann. Regulation of synaptic efficacy by coincidence of postsynaptic APs and EPSPs. *Science*, 275:213–215, 1997.

[14] B. Porr and F. Wörgötter. Learning with "relevance": Using a third factor to stabilise hebbian learning. *Neural Comput.*, 19:2694–2719, 2007.

[15] C. Watkins and P. Dayan. Technical note:Q-Learning. *Mach. Learn.*, 8:279–292, 1992.

[16] S. P. Singh, T. Jaakkola, M. L. Littman, and C. Szepesvári. Convergence results for single-step on-policy reinforcement-learning algorithms. *Mach. Learn.*, 38(3):287–308, 2000.

[17] W. Gerstner, R. Kempter, L. van Hemmen, and H. Wagner. A neuronal learning rule for sub-millisecond temporal coding. *Nature*, 383:76– 78, 1996.

[18] R. Rao and T. Sejnowski. Spike-timing-dependent hebbian plasticity as temporal difference learning. *Neural Comput.*, 13:2221–2237, 2001.

[19] J. Baxter, P. L. Bartlett, and L. Weaver. Experiments with infinite-horizon,policy-gradient estimation. *J. Artif. Intell. Res.*, 15:351–381, 2001.

[20] W. Schultz, P. Apicella, E. Scarnati, and T. Ljungberg. Neuronal activity in monkey ventral striatum related to the expectation of reward. *J. Neurosci.*, 12(12):4595–610, 1992.

[21] P. R. Montague, P. Dayan, and T. J. Sejnowski. A framework for mesencephalic dopamine systems based on predictive hebbian learning. *J. Neurosci.*, 76(5):1936–1947, 1996.

[22] G. Morris, A. Nevet, D. Arkadir, E. Vaadia, and H. Bergman. Midbrain dopamine neurons encode decisions for future action. *Nat. Neurosci.*, 9 (8):1057–1063, 2006.

[23] P. Dayan. Matters temporal. *Trends. Cogn. Sci.*, 6(3):105–106, 2002.

[24] C. Kolodziejski, B. Porr, and F. Wörgötter. Mathematical properties of neuronal TD-rules and differential hebbian learning: A comparison. *Biol. Cybern.*, 98(3):259–272, 2008.
